# Hidden Markov Models in Molecular Biology: New Algorithms and Applications

**Pierre Baldi** [*]
Jet Propulsion Laboratory
California Institute of Technology
Pasadena, CA 91109

**Yves Chauvin** [†]
Net-ID, Inc.
8, Cathy Place
Menlo Park, CA 94305

**Tim Hunkapiller**
Division of Biology
California Institute of Technology

**Marcella A. McClure**
Department of Evolutionary Biology
University of California, Irvine

## Abstract

Hidden Markov Models (HMMs) can be applied to several important problems in molecular biology. We introduce a new convergent learning algorithm for HMMs that, unlike the classical Baum-Welch algorithm is smooth and can be applied on-line or in batch mode, with or without the usual Viterbi most likely path approximation. Left-right HMMs with insertion and deletion states are then trained to represent several protein families including immunoglobulins and kinases. In all cases, the models derived capture all the important statistical properties of the families and can be used efficiently in a number of important tasks such as multiple alignment, motif detection, and classification.

[*]and Division of Biology, California Institute of Technology.
[†]and Department of Psychology, Stanford University.

## 1   INTRODUCTION

Hidden Markov Models (e.g., Rabiner, 1989) and the more general EM algorithm in statistics can be applied to the modeling and analysis of biological primary sequence information (Churchill (1989), Lawrence and Reilly (1990), Baldi et al. (1992), Cardon and Stormo (1992), Haussler et al. (1992)). Most notably, as in speech recognition applications, a family of evolutionarily related sequences can be viewed as consisting of different utterances of the same prototypical sequence resulting from a common underlying HMM dynamics. A model trained from a family can then be used for a number of tasks including multiple alignments and classification. The multiple alignment is particularly important since it reveals the highly conserved regions of the molecules with functional and structural significance even in the absence of any tertiary information. The multiple alignment is also an essential tool for proper phylogenetic tree reconstruction and other important tasks. Good algorithms based on dynamic programming exist for the alignment of two sequences. However they scale exponentially with the number of sequences and the general multiple alignment problem is known to be NP-complete. Here, we briefly present a new algorithm and its variations for learning in HMMs and the results of some of the applications of this approach to new protein families.

## 2   HMMs FOR BIOLOGICAL PRIMARY SEQUENCES

A HMM is characterized by a set of states, an alphabet of symbols, a probability transition matrix $T = (t_{ij})$ and a probability emission matrix $e_{ij}$. As in speech applications, we are going to consider left-right architectures: once a given state is left it can never be visited again. Common knowledge of evolutionary mechanisms suggests the choice of three types of states (in addition to the start and to the end state): the main states $m_1, ..., m_N$, the delete states $d_1, ..., d_{N+1}$ and the insert states $i_1, ..., i_{N+1}$. $N$ is the length of the model which is usually chosen equal to the average length of the sequences in the family and, if needed, can be adjusted in later stages. The details of a typical architecture are given in Figure 1. The alphabet has 4 letters in the case of DNA or RNA sequences, one symbol per nucleotide, and 20 letters in the case of proteins, one symbol per amino acid. Only the main and insert states emit letters, while the delete states are of course mute. The linear sequence of state transitions $start \rightarrow m_1 \rightarrow m_2 \rightarrow ... \rightarrow m_N \rightarrow end$ is the backbone of the model and correponds to the path associated with the prototypical sequence in the family under consideration. Insertions and deletions are defined with respect to this backbone. Insertions and deletions are treated symmetrically except for the loops on the insert states needed to account for multiple insertions. The adjustable parameters of the HMM provide a natural way of incorporating variable gap penalties. A number of other architectures are also possible.

## 3   LEARNING ALGORITHMS

Learning from examples in HMMs is typically accomplished using the Baum-Welch algorithm. In the Baum-Welch algorithm, the expected number $n_{ij}$ (resp. $m_{ij}$) of $i \rightarrow j$ transitions (resp. emissions of letter $j$ from state $i$) induced by the data are calculated using the forward-backward procedure. The transition and emission

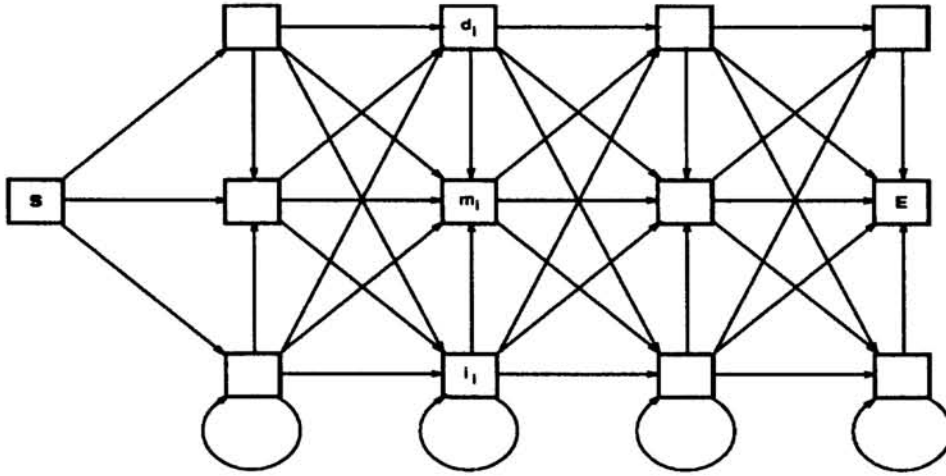

Figure 1: The basic left-right HMM architecture. S and E are the start and end states.

probabilities are then reset to the observed frequencies by

$$t_{ij}^+ = \frac{n_{ij}}{n_i} \quad \text{and} \quad e_{ij}^+ = \frac{m_{ij}}{m_i} \tag{1}$$

where $n_i = \sum_j n_{ij}$ and $m_i = \sum_j m_{ij}$. It is clear that this algorithm can lead to abrupt jumps in parameter space and that the procedure cannot be used for on-line learning (after each training example). This is even more so if, in order to save some computations, the Viterbi approximation is used to estimate likelihoods and transition and emission statistics by computing only the most likely paths as opposed to the forward-bacward procedure where all possible paths are examined.

A new algorithm for HMM learning which is smooth and can be used on-line or in batch mode, with or without the Viterbi approximation, can be defined as follows. First, we use a Boltzmann-Gibbs representation for the parameters. For each $t_{ij}$ (resp. $e_{ij}$) we define a new parameter $w_{ij}$ (resp. $v_{ij}$) by

$$t_{ij} = \frac{e^{w_{ij}}}{\sum_k e^{w_{ik}}} \quad \text{and} \quad e_{ij} = \frac{e^{v_{ij}}}{\sum_k e^{v_{ik}}} \tag{2}$$

Normalisation constraints are naturally enforced by this representation throughout learning with the added advantage that none of the parameters can reach the absorbing value 0. After computing on-line or in batch mode the statistics $n_{ij}$ and $m_{ij}$ using the forward-backward procedure (or the usual Viterbi approximation), the update equations are particularly simple and given by

$$\Delta w_{ij} = \eta(\frac{n_{ij}}{n_i} - t_{ij}) \quad \text{and} \quad \Delta v_{ij} = \eta(\frac{m_{ij}}{m_i} - e_{ij}) \tag{3}$$

where $\eta$ is the learning rate. In Baldi et al. (1992) a proof is given that this algorithm must converge to a maximum of the product of the likelihoods of the training sequences. In the case of an on-line Viterbi approximation, the optimal path associated with the current training sequence is first computed. The update equations are then given by

$$\Delta w_{ij} = \eta(t_{ij}^t - t_{ij}) \quad \text{and} \quad \Delta v_{ij} = \eta(t_{ij}^e - e_{ij}) \tag{4}$$

Here, for a fixed state $i$, $t_{ij}^t$ and $t_{ij}^e$ are the target transition and emission values: $t_{ij}^t = 1$ every time the transition $s_i \rightarrow s_j$ is part of the Viterbi path of the corresponding training sequence sequence and 0 otherwise and similarly for $t_{ij}^e$.

After training, the model derived can be used for a number of tasks. First, by computing for each sequence its most likely path through the model using the Viterbi algorithm, multiple sequences can be aligned to each other in time $O(KN^2)$, linear in the number $K$ of sequences. The model can also be used for classification and data base searches. The likelihood of any sequence (randomly generated or taken from any data base) can be calculated and compared to the likelihood of the sequences in the family being modeled. Additional applications are discussed in Baldi et al. (1992).

## 4 EXPERIMENTS AND RESULTS

The previous approach has been applied to a number of protein families including globins, immunoglobulins, kinases, aspartic acid proteases and G-coupled receptor proteins. The first application and alignment of the globin family using HMMs (trained with the Viterbi approximation of the Baum-Welch algorithm, and a number of additional heuristics) was given by Haussler et al. (1992). Here, we briefly describe some of our results on the immunoglobulin and the kinase families [1].

### 4.1 IMMUNOGLOBULINS

Immunoglobulins or antibodies are proteins produced by B cells that bind with specificity to foreign antigens in order to neutralize them or target their destruction by other effector cells (e.g., Hunkapiller & Hood, 1989). The set of sequences used in our experiments consists of immunoglobulins V region sequences from the Protein Identification Resources (PIR) data base. It corresponds to 294 sequences, with minimum length 90, average length 117 and maximum length 254. The variation in length resulted from including any sequence with a V region, including those that also included signal or leader sequences, germline sequences that did not include the J segment, and some that contained the C region as well. Seventy seqences contained one or more special characters indicating an ambiguous amino acid determination and were removed.

For the immunoglobulins variable V regions, we have trained a model of length 117 using a random subset of 150 sequences. Figure 2 displays the alignment corresponding to the first 20 sequences in this random subset. Letters emitted from the main states are upper case and letters emitted from insertion states are lower case. Dashes represent deletions or accomodate for insertions. As can be observed, the algorithm has been able to detect all the main regions of highly conserved residues. Most importantly, the cysteine residues towards the beginning and the end responsible for the disulphide bonds which holds the chains together are perfectly aligned and marked. The only exception is the fifth sequence from the bottom which has a serine residue in its terminal portion. It is also important to remark that some

```
PH0106  mklpvrllvlmfwipasssDvVMTQTPLSLpvSLGDQASISCRSSQSLVHSngnTYLNWYLQ--KAGQS-
B27563  ---------------------LQQPGAELv-KPGASVKLSCKASGYTFTN---YWIHWVKQ--RPGRGL
MHMS76  --------------------ESGGGLv-QPGGSMKLSCVASGFTFSN---YWMNWVRQ--SPEKGL
D28035  mefglswiflvailkgvqcEvRLVESGGDLv-EPGGSLRVSCEVSGFIFSK---AWMNWVRQ--APGKGL
D24672  -----------------------------------ISCKASGYTFTN---YGMNWVKQ--APGKGL
PH0100  -------------------LvQLQQSGPVLv-KPGTSMKISCKTSGYSFTG---YTMSWVRQ--SHGKSL
B27888  -------------------EvMLVESGGGLa-KPGGSLKLSCTTSGFTFSI---HAMSWVRQ--TPEKRL
PL0160  -------------------QvQLQQSGPGLv-KPSQTLSLTCAISGDSVSSns-AAWNWIRQ--SPSRGL
E28833  -------------------DvVMTQTPLSLpvSLGDQASISCRSSQSLVRSngnTYLHWYLQ--KPGQP-
D30539  -------------------EvKLVESGGGLv-QSGGSLRLSCATSGFTFSD---FYMEWVRQ--PPGKSL
C30560  -------------------QvHLQQSGAELv-KPGASVKISCKASGYTFTS---YWMNWVKQ--RPGQGL
AVMSX4  -------------------EvKLLESGGGLv-QPGGSLKLSCAASGFDFSR---YWMSWVRQ--APGKGL
C30540  -------------------EvKLVESGGGLv-QPGGSLRLSCATSGFTFSD---FYMEWVRQ--PPGKRL
PL0123  -------------------EvQLVESGGGLv-QPGGSLRLSCAASGFTFSS---YWMSWVRQ--APGKGL
H36005  -------------------EvQLVESGGGLv-KPGGSLRLSCAASGFTFSN---AWMNWVRQ--APGKGL
PH0097  -------------------DvKLVESGGGLv-KPGGSLKLSCAASGFTFSS---YIMSWVRQ--TPEKRL
I37267  gsimg--------------vQLQQSGPELv-KPGASVKISCKTSGYTFTE---YTMHWVKQ--SHGKSL
A25114  -------------------DvHLQESGPGLv-KPSQSLSLTCSVTGYSITRg--YNWNWIRR--FPGNKL
D2HUWA  -------------------RlQLQESGPGLv-KPSETLSLTCIVSGGPIRRtg-YYWGWIRQ--PPGKGL
A30539  -------------------EvKLVESGGGLv-QPGGSLRLSCATSGFTFSD---FYMEWVRQ--PPGKRL
........................................*............................

PH0106  -p-KLLI-YKV---SNR-FSGVPDRFSGSG--SGTDFTLKISRVEAEDLGIYFCSQ---------------
B27563  E-WIGRI-DPNSGGTKY-NEKFKNKATLTINKPSNTAYMQLSSLTSDDSAVYYCARGYDYSYY-------
MHMS76  E-WVAEIrLKSGYATHY-AESVKGRFTISRDDSKSSVYLQMNNLRAEDTGIYYCTRPGV----------
D28035  Q-WVGQIkNKVDGGTIDYAAPVKGRFIISRDDSKSTVYLQMNRLKIEDTAVYYCVGNYTGT---------
D24672  K-WMGWI-NTYTGEPTY-ADDFKGRFAFSLETSASTAYLQINNLKNEDTATYFCARGSSYDYY-------
PH0100  E-WIGLI-IPSNGGTNY-NQKFKDKASLTVDKSSSTAYMELLSLTSEDSAVYYCARPSYYGSRnyy----
B27888  E-WVAAI-SSGGSYTFY-PDSVKGRFTISRDNAKNTLYLQINSLRSEDTAIYYCAREEGLRLDdy-----
PL0160  E-WLGRT-YYRSKWYNDYAVSVKSRITINPDTSKNQFSLQLNSVTPEDTAVYYCARELGDA---------
E28833  -p-KLLI-YKV---SNR-VSGVPDRFSGSG--SGTDFTLKISRVEAEDLGVYFCSQSTHV----------
D30539  E-WIAASrNEANDYTTEYSASVKGRFIVSRDTSQSILYLQMIALRAEDTAIYYCSRDYYGSSYw------
C30560  E-WIGEI-DPSNSYTNN-NQKFKNKATLTVDKSSNTAYMQLSSLTSEDSAVYYCARWGTGSSWg------
AVMSX4  E-WIGEI-NPDSSTINY-TPSLKDKFIISRDNAKNTLYLQMSKVRSEDTALYYCARLHYYGY--------
C30540  E-WIAASrNKAHDYTTEYSASVKGRFIVSRDTSQSILYLQMNALRAEDTAIYYCARDADYGSSshw----
PL0123  E-WVANI-KQDGSEKYY-VDSVKGRFTISRDNAKNSLYLQMNSLRAEDTAVYYCAR-------------
H36005  E-WVGRIkSKTDGGTTDYAAPVKGRFTISRDDSKNTLYLQMNSLKTEDTAVYYCTTDRGGSSQ-------
PH0097  E-WVATI-SSGGRYTYY-SDSVKGRFTISRDNAKNTLYLQMSSLRSEDTAMYYSTASGDS----------
I37267  E-WIGGI-NPNNGGTSY-NQKFKGKATLTVDKSSSTAYMELRSLTSEDSAVYYCARRGLTTVVaksy---
A25114  E-WMGYI-NYDGS-NNY-NPSLKNRISVTRDTSKNQFFLKMNSVTTEDTATYYCARLIPFSDGyyedyy-
D2HUWA  E-WIGGV-YYTGS-IYY-NPSLRGRVTISVDTSRNQFSLNLRSMSAADTAMYYCARGNPPPYYdigtgsd
A30539  E-WIAASrNKANDYTTEYSASVKGRFIVSRDTSQSILYLQMNALRAEDTAIYYCARDYYGSSYvw-----
..................................................*..................

PH0106  ----------------tthvpptfgggtkleikr-
B27563  -AMDYWGQGTSVTVSS-------------------
MHMS76  --PDYWGQGTTLTVSS-------------------
D28035  --VDYWGQGTLVTVSS-------------------
D24672  -AMDYWGQGTSVTVSS-------------------
PH0100  -AMDYWGQGTSVTVSSak-----------------
B27888  -AMDYWGQGTSVTVS--------------------
PL0160  --FDIWGQGTMVTVSS-------------------
E28833  -----------------------------------
D30539  -YFDVWGAGTTVTVSS-------------------
C30560  -WFAYWGQGTLVTVSA-------------------
AVMSX4  --AAYWGQGTLVTVSAe------------------
C30540  -YFDVWGAGTTVTVSS-------------------
PL0123  -----------------------------------
H36005  --GDYWGQGTLVTVSS-------------------
PH0097  --FDYWGQGTTLTVSSak-----------------
I37267  -YFDYWGQGTTLTVSS-------------------
A25114  -AMDYWGQGT-------------------------
D2HUWA  dGIDVWGQGTTVHVSS-------------------
A30539  -YFDVWGAGTTVTVSS-------------------
...............................................
```

Figure 2: Immunoglobulin alignment.

of the sequences in the family have some sort of "header" (leader signal peptide) whereas the others do not. We did not remove the headers prior to training and used the sequences as they were given to us. The model was able to detect and accomodate these "headers" by treating them as initial inserts as can be seen from the alignment of two of the sequences.

## 4.2 KINASES

Eukaryotic protein kinases constitute a very large family of proteins that regulate the most basic of cellular processes through phosphorylation. They have been termed the "transistors" of the cell (Hunter (1987)). We have used the sequences available in the kinase data base maintained at the Salk Institute. Our basic set consists of 224 sequences, with minimum length 156, average length 287, and maximal length 569. Only one sequence containing a special symbol (X) was discarded. In one experiment, we trained a model of length 287 using a random subset of 150 kinase sequences. Figure 3 displays the corresponding alignment for a subset of 12 phylogenetically representative sequences. These include serine/threonine, tyrosine and dual specificity kinases from mammals, birds, fungi and retroviruses and herpes viruses. The percentage of identical residues within the kinase data sets ranges from 8-30%, suggesting that only those residues involved in catalysis are conserved among these highly divergent sequences. All the 12 characteristic catalytic domains or subdomains described in Hanks and Quinn (1991) are easily recognizable and marked. Additional highly conserved positions can also be observed consistent with previously constructed multiple alignments. For instance, the initial hydrophobic consensus Gly-X-Gly-XX-Gly together with the Lys located 15 or 20 residues downstream are part of the ATP/GTP binding site. The carboxyl terminus is characterized by the presence of an invariant Arg residue. Conserved residues in proximity to the acceptor amino acid are found in the VIb (Asp), VII (Asp-Phe-Gly) and VIII domains (Ala-Pro-Glu). In Figure 4, the entropy of the emission distribution of each main state is plotted: motifs are easily detectable and correspond to positions with very low entropy.

## 5  DISCUSSION

HMMs are emerging as a powerful, adaptive, and modular tool for computational biology. Here, they have been used, together with a new learning algorithm, to model families of proteins. In all cases, the models derived capture all the important statistical properties of the families. Additional results and potential applications, such as phylogenetic tree reconstruction, classification, and superfamily modeling, are discussed in Baldi et al. (1992).

## Footnotes

[1] Recently, Hausssler et al. have also independently applied their approach to the kinase family (Haussler, private communication).

## References

Baldi, P., Chauvin, Y., Hunkapiller, T. and McClure, M. A. (1992) Adaptive Algorithms for Modeling and Analysis of Biological Primary Sequence Information. Technical Report.

Cardon, L. R. and Stormo, G. D. (1992) Expectation Maximization Algorithm for Identifying Protein-binding Sites with Variable Lengths from Unaligned DNA

```
CD28    anYKR--LEKVGEGTYGVVYKALDLrpg--QGQRVVALK------KIRLESEDEGVPSTAIREISLLKEL-K-DDNIVRLYDIVH
MLCK    --FSMnsKEALGGGKFGAVCTCTEK-----STGLKLAAK---VI-KKQTPKDKE----MVMLEIEVMNQL-N-HRNLIQLYAAIE
PSKH    akYDI--KALIGRGSFSRVVRVEHR-----ATRQPYAIK---MIETKYREGRE-----VCESELRVLRRV-R-HANIIQLVEVFE
CAPK    dqFER--IKTLGTGSFGRVMLVKHM-----ETGNHYAMK---ILDKQKVVKLKQIE--HTLNEKRILQAV-N-FPFLVKLEFSFK
WEE1    trFRN--VTLLGSGEFSEVFQVEDPv----EKTLKYAVK---KL-KVKFSGPKERN--RLLQEVSIQRALkG-HDHIVELMDSWE
CSRC    esLRL--EVKLGQGCFGEVWMGTWN------GTTRVAIK---TLKPGNMSPE------AFLQEAQVMKKL-R-HEKLVQLYAVVS
EGFR    teFKK--IKVLGSGAFGTVYKGLWIpege-KVKIPVAIK---ELREATSPKANK----EILDEAYVMASV-D-NPHVCRLLGICL
PDGF    dqLVL--GRTLGSGAFGQVVEATAHGlshsQATMKVAVK---MLKSTARSSEKQ----ALMSELY--GDL--v-DYLHRNKHTFL
VFES    edLVL--GEQIGRGNFGEVFSGRLR-----ADNTLVAVK---SCRETLPPDIKA----KFLQEAKILKQY-S-HPNIVRLIGVCT
RAF1    seVML--STRIGSGSFGTVYKGKWH--------GDVAVK---ILKVVDPTPEQFQ---AFRNEVAVLRKT-R-HVNILLFMGYMT
CMOS    eqVCL--LQRLGAGGFGSVYKATYR-------GVPVAIKQvNKCTKNRLASRR-----SFWAELNV-ARL-R-HDNIVRVVAAST
HSVK    mgFTI--HGALTPGSEGCVFDSSHP-----DYPQRVIVK------AGWYT--------STSHEARLLRRL-D-HPAILPLLDLHV
........................*...*.*.........................................*.................
.......................I.............II.............III...........IV.....
```

```
CD28    SDAHk---------LY-L-V-FEFLDL-DLKRYMEGIpkd------------------------------
MLCK    TPHE----------IV-L-F-MEYIEGGELFERIVDE-------------------------------
PSKH    TQER----------VY-M-V-MELATGGELFDRIIAK-------------------------------
CAPK    DNSN----------LY-M-V-MEYVPGGEMFSHLRRI-------------------------------
WEE1    HGGF----------LY-M-Q-VELCENGSLDRFLEEQgql---------------------------
CSRC    -EEP----------IY-I-V-TEYMSKGSLLDFLKGE------------------------------
EGFR    -TST----------VQ-L-I-TQLMPFGCLLDYVREH------------------------------
PDGF    -QRHsnkhcppsaeLYs-n-a--LPVGFSLPSHLNLTgesdggymdmskdesidyvpmldmkgdikyadiespsymapydnyvps
VFES    QKQP----------IY-V-M-ELVQGGDFLTFLRTE-------------------------------
RAF1    -KDN----------LA-I-V-TQWCEGSSLYKHLHVQ------------------------------
CMOS    RTPAgsnsl-----GT-I-I-MEFGGNVTLHQVIYGAaghpegdagephcrtg--------------
HSVK    VSGV----------TC-L-V-LPKYQA-DLYTYLSRR-------------------------------
.............................................V.............................
```

```
CD28    ----------------QP-LGADIVKKFMMQ-LCKGIAYCHSHRILHRDLKPQNLL-INKDG---N-LKLGDFGLARAFGVPLRAY
MLCK    --------------DYH-LTEVDTMVFVRQ-ICDGILFMHKMRVLHLDLKPENILcVNTTG---HlVKIIDFGLARRYNPNEKL-
PSKH    ---------------GS-FTERDATRVLQM-VLDGVRYLHALGITHRDLKPENLL-YYHPGtdsK-IIITDFGLASARKKGDDCL
CAPK    ---------------GR-FSEPHARFYAAQ-IVLTFEYLHSLDLIYRDLKPENLL-IDQQG---Y-IQVTDFGFAKRVKGRT---
WEE1    ---------------SR-LDEFRVWKILVE-VALGLQFIHHKNYVHLDLKPANVM-ITFEG---T-LKIGDFGMASVWPVPRG--
CSRC    --------------MGKyLRLPQLVDMAAQ-IASGMAYVERMNYVHRDLRAANIL-VGENL---V-CKVADFGLARLIEDNEYTA
EGFR    --------------KDN-IGSQYLLNWCVQ-IAKGMNYLEDRRLVHRDLAARNVL-VKTPQ---H-VKITDFGLAKLLGAEEKEY
PDGF    apertyratlinds-PV-LSYTDLVGFSYQ-VANGMDFLASKNCVHRDLAARNVL-ICEGK---L-VKICDFGLARDIMRDSNYI
VFES    --------------GAR-LRMKTLLQMVGD-AAAGMEYLESKCCIHRDLAARNCL-VTEKN---V-LKISDFGMSREAADGIYAA
RAF1    --------------ETK-FQMFQLIDIARQ-TAQGMDYLHAKNIIHRDMKSNNIF-LHEGL---T-VKIGDFGLATVKSRWSGSQ
CMOS    --------------GQ-LSLGKCLKYSLD-VVNGLLFLHSQSIVHLDLKPANIL-ISEQD---V-CKISDFGCSEKLEDLLCFQ
HSVK    --------------LNP-LGRPQIAAVSRQ-LLSAVDYIHRQGIIHRDIKTENIF-INTPE---D-ICLGDFGAACFVQGSRSSP
.................................................*....*........*.......*.*.*..........
....................VIa............VIb..............VII...........
```

```
CD28    ---THEIVTLWYRAPEVLLggGK---QYSTGVDTWSIGCIFAEMCNRKP---------------IFSGDSE-----IDQIFKIFRV
MLCK    ---KVNFGTPEFLSPEVVN-YD---QISDKTDMWSLGVITYMLLSGLS---------------PFLGDDD-----TETLNNVLSG
PSKH    M--KTTCGTPEYIAPEVLV-RK---PYTNSVDMWALGVIAYILLSGTM---------------PFEDDNR-----TRLYRQILRG
CAPK    ---WTLCGTPEYLAPEIIL-SK---GYNKAVDWWALGVLIYEMAAGYP---------------PFFADQP-----IQIYEKIVSG
WEE1    ---MEREGDCEYIAPEVLA-NH---LYDKPADIFSLGITVFEAAANIV--------------LPDNGQSW-----Q----KLRSG
CSRC    R--QGAKFPIKWTAPEAAL-YG---RFTIKSDVWSFGILLTELTTKGR-------------VPYPGMVN-----REVLDQVERG
EGFR    H--AEGGKVPIKWMALESIL-HR---IYTHQSDVWSYGVTVWELMTFGS--------------KPYDGIPA-----SEISSILEKG
PDGF    S--KGSTYLPLKWMAPESIF-NS---LYTTLSDVWSFGILLWEIFTLGG-------------TPYPELPM----NDQFYNAIKRG
VFES    S--GGLRQVPVKWTAPEALN-YG---RYSSESDVWSFGILLWETFSLGA-------------SPYPNLSN-----QQTREFVEKG
RAF1    Q--VEQPTGSVLWMAPEVIR-MQdnnPFSFQSDVYSYGIVLYELMTGEL--------------PYS---R-----DQIIFMVGRG
CMOS    TpSYPLGGTYTHRAPELLK-GE---GVTPKADIYSFAITLWQMTTKQA--------------PYSGERQ-----HILYAVVAYD
HSVK    F--PYGIAGTIDTNAPEVLA-GD---PYTTTVDIWSAGLVIFETAVHNA---------------------------
....................**.......*.....*........................................
....................VIII........IX....................X........
```

```
CD28    ---LGTPNEAIwpdivylpdfkpsfpqwrrkdlsqvvpsLDPRGIDLLDKLLAYDPiNRISARRAAIHPYFQES--------
MLCK    nwyFDEETFEA----------------------------VSDEAKDFVSNLIVKEQGARMSAAQCLAHPWLNNL--------
PSKH    kysYSGEPWPS----------------------------VSNLAKDFIDRLLTVDPGARMTALQALRHPWVVSM--------
CAPK    ---KVR-FPSH----------------------------FSSDLKDLLRNLLQVDLTKRFGNLKDGVNDIKNHK--------
WEE1    ---DLSDAPRLsstdngssltsssretpansii------GQGGLDRVVEWMLSPEPRNRPTIDQILATD--EVCWV------
CSRC    ---YRMPCPPE----------------------------CPESLHDLMCQCWRRDPEERPTFEYLQAFLEDYFT--------
EGFR    ---ERLPQPPI----------------------------CTIDVYMIMVKCWMIDADSRPKFRELIIEFSKMAR--------
PDGF    ---YRMAQPAH----------------------------ASDEIYEIMQKCWEEKFETRPPFSQLVLLLERLLGEGykkky-
VFES    ---GRLPCPEL----------------------------CPDAVFRLMEQCWAYEPGQRPSFSAIYQEL-------------
RAF1    ---YASPDLSKlykn------------------------CPKAMKRLVADCVKKVKEERPLFPQILSSIELLQH--------
CMOS    ---LRPSLSAAvfedsl----------------------PGQRLGDVIQRCWRPSAAQRPSARLLLVDLTSLKA--------
HSVK    -------------------------------------------------------------------------------
.....................................................................*...........
..............................................XI..........
```

Figure 3: Kinase alignment of 12 representative sequences.

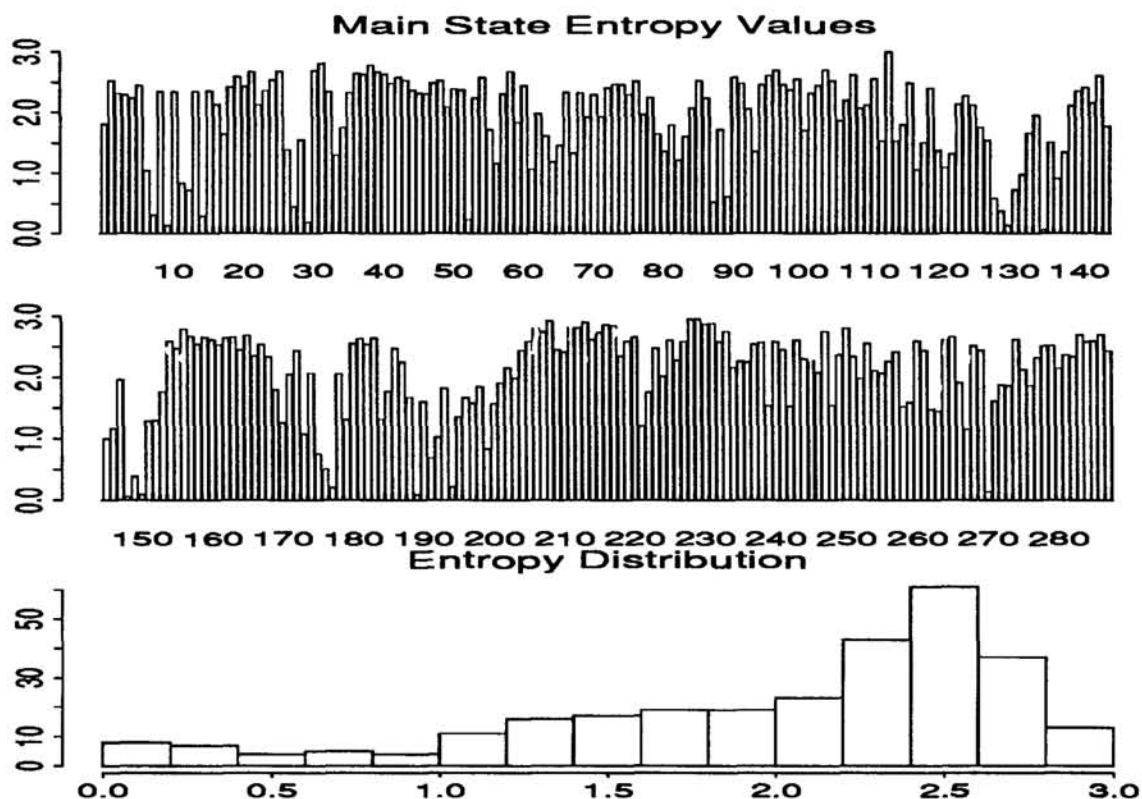

Figure 4: Kinase emission entropy plot and distribution.

Fragments. Journal of Molecular Biology, **223**, 159-170.

Churchill, G. A. (1989) Stochastic Models for Heterogeneous DNA Sequences. Bulletin of Mathematical Biology, **51**, 1, 79-94.

Hanks, S. K., Quinn, A. M. (1991) Protein Kinase Catalytic Domain Sequences Database: Identification of Conserved Features of Primary Structure and Classification of Family Members. Methods in Enzymology, **200**, 38-62.

Haussler, D., Krogh, A., Mian, S. and Sjolander, K. (1992) Protein Modeling using Hidden Markov Models. Computer and Information Sciences Technical Report (UCSC-CRL-92-93), University of California, Santa Cruz.

Hunkapiller, T. and Hood, L. (1989) Diversity of the Immunoglobulin Gene Superfamily. Advances in Immunology, **44**, 1-63, Academic Press, Inc.

Hunter, T. (1987) A Thousand and One Protein Kinases. Cell, **50**, 823-829.

Lawrence, C. E. and Reilly, A. A. (1990) An Expectation Maximization (EM) Algorithm for the Identification and Characterization of Common Sites in Unaligned Biopolymer Sequences. Proteins: Struct. Funct. Genet., **7**, 41-51.

Rabiner, L. R. (1989) A Tutorial on Hidden Markor Models and Selected Applications in Speech Recognition. Proceedings of the IEEE, **77**, 2, 257-286.
